# Continuously-adaptive discretization for message-passing algorithms

**Kannan Achan**
Microsoft Research Silicon Valley
Mountain View, California, USA

**Michael Isard**
Microsoft Research Silicon Valley
Mountain View, California, USA

**John MacCormick**
Dickinson College
Carlisle, Pennsylvania, USA

## Abstract

Continuously-Adaptive Discretization for Message-Passing (CAD-MP) is a new message-passing algorithm for approximate inference. Most message-passing algorithms approximate continuous probability distributions using either: a family of continuous distributions such as the exponential family; a particle-set of discrete samples; or a fixed, uniform discretization. In contrast, CAD-MP uses a discretization that is (i) non-uniform, and (ii) adaptive to the structure of the marginal distributions. Non-uniformity allows CAD-MP to localize interesting features (such as sharp peaks) in the marginal belief distributions with time complexity that scales logarithmically with precision, as opposed to uniform discretization which scales at best linearly. We give a principled method for altering the non-uniform discretization according to information-based measures. CAD-MP is shown in experiments to estimate marginal beliefs much more precisely than competing approaches for the same computational expense.

## 1   Introduction

Message passing algorithms such as Belief Propagation (BP) [1] exploit factorization to perform inference. Exact inference is only possible when the distribution to be inferred can be represented by a tree and the model is either linear-Gaussian or fully discrete [2, 3]. One attraction of BP is that algorithms developed for tree-structured models can be applied analogously [4] to models with loops, such as Markov Random Fields.

There is at present no general-purpose approximate algorithm that is suitable for all problems, so the choice of algorithm is governed by the form of the model. Much of the literature concentrates on problems from statistics or control where point measurements are made (e.g. of an animal population or a chemical plant temperature), and where the state evolution is non-linear or the process noise is non-Gaussian [5, 6]. Some problems, notably those from computer vision, have more complex observation distributions that naturally occur as piecewise-constant functions on a grid (i.e. images), and so it is common to discretize the underlying continuous model to match the structure of the observations [7, 8]. As the dimensionality of the state-space increases, a naïve uniform discretization rapidly becomes intractable [8]. When models are complex functions of the observations, sampling methods such as non-parametric belief propagation (NBP) [9, 10], have been successful.

Distributions of interest can often be represented by a factor graph [11]. "Message passing" is a class of algorithms for approximating these distributions, in which messages are iteratively updated between factors and variables. When a given message is to be updated, all other messages in the graph are fixed and treated as though they were exact. The algorithm proceeds by picking, from

a family of approximate functions, the message that minimizes a divergence to the local "exact" message. In some forms of the approach [12] this minimization takes place over approximate belief distributions rather than approximate messages.

A general recipe for producing message passing algorithms, summarized by Minka [13], is as follows: (i) pick a family of approximating distributions; (ii) pick a divergence measure to minimize; (iii) construct an optimization algorithm to perform this minimization within the approximating family. This paper makes contributions in all three steps of this recipe, resulting in a new algorithm termed *Continuously-Adaptive Discretization for Message-Passing* (CAD-MP).

For step (i), we advocate an approximating family that has received little attention in recent years: piecewise-constant probability densities with a bounded number of piecewise-constant regions. Although others have used this family in the past [14], it has not to our knowledge been employed in a modern message-passing framework. We believe piecewise-constant probability densities are very well suited to some problem domains, and this constitutes the chief contribution of the paper. For step (ii), we have chosen for our initial investigation the "inclusive" KL-divergence [13]—a standard choice which leads to the well known Belief Propagation message update equations. We show that for a special class of piecewise-constant probability densities (the so-called *naturally-weighted* densities), the minimal divergence is achieved by a distribution of minimum entropy, leading to an intuitive and easily-implemented algorithm. For step (iii), we employ a greedy optimization by traversing axis-aligned binary-split kd-trees (explained in Section 3). The contribution here is an efficient algorithm called "informed splitting" for performing the necessary optimization in practice.

As we show in Section 4, CAD-MP computes much more accurate approximations than competing approaches for a given computational budget.

## 2 Discretizing a factor graph

Let us consider what it means to *discretize* an inference problem represented by a factor graph with factors $f_i$ and continuous variables $x_\alpha$ taking values in some subset of $\mathbb{R}^N$. One constructs a non-uniform discretization of the factor graph by partitioning the state space of each variable $x_\alpha$ into $K$ regions $H_\alpha^k$ for $k = 1, \ldots, K$. This discretization induces a discrete approximation $f_i'$ of the factors, which are now regarded as functions of discrete variables $x_\alpha'$ taking integer values in the set $\{1, 2, \ldots, K\}$:

$$f_i'(k, l, \ldots) = \int_{x_\alpha \in H_\alpha^k, x_\beta \in H_\beta^l, \ldots} f_i(x_\alpha, x_\beta, \ldots), \tag{1}$$

for $k, l, \ldots = 1, \ldots, K$. A slight variant of BP [4] could then be used to infer the marginals on $x_\alpha'$ according to the update equations for messages $m$ and beliefs $b$:

$$m_{\alpha,i}(k) = \prod_{f_j' \sim x_\alpha' \setminus f_i'} m_{j,\alpha}(k) \tag{2}$$

$$m_{i,\alpha}(k) = \frac{1}{|H_\alpha^k|} \sum_{\mathbf{x}'|x_\alpha'=k} f_i'(\mathbf{x}') \prod_{x_\beta' \sim f_i' \setminus x_\alpha'} m_{\beta,i}(x_\beta') \tag{3}$$

$$b_\alpha(k) = |H_\alpha^k| \prod_{f_j' \sim x_\alpha'} m_{i,\alpha}(k), \tag{4}$$

where $a \sim b \setminus c$ means "all neighbors $a$ of $b$ except $c$", $\mathbf{x}'$ is an assignment of values to all variables, and $|H_\alpha^k| = \int_{H_\alpha^k} 1$. Thus, given a factor graph of continuous variables and a particular choice of discretization $\{H_\alpha^k\}$, one gets a piecewise-constant approximation to the marginals by first discretizing the variables according to (1), then using BP according to (2)–(4). The error in the approximation to the true marginals arises from (3) when $f_i'(\mathbf{x})$ is not constant over $\mathbf{x}$ in the given partition.

Consider the task of selecting between discretizations of a continuous probability distribution $p(x)$ over some subset $U$ of Euclidean space. A *discretization* of $p$ consists in partitioning $U$ into $K$ disjoint subsets $V_1, \ldots, V_K$ and assigning a weight $w_k$ to each $V_k$, with $\sum_k w_k = 1$. The corresponding discretized probability distribution $q(x)$ assigns density $w_k/|V_k|$ to $V_k$. We are interested in finding a discretization for which the KL divergence $\mathrm{KL}(p||q)$ is as small as possible. The optimal choice of the $w_k$ for any fixed partitioning $V_1, \ldots, V_K$ is to take $w_k = \int_{x \in V_k} p(x)$ [14]; we call

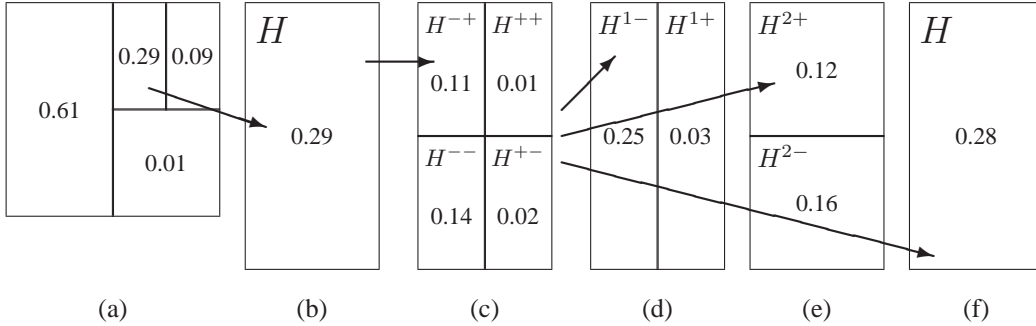

Figure 1: **Expanding a hypercube in two dimensions.** Hypercube $H$ (b), a subset of the full state space (a), is first "expanded" into the sub-cubes $\{H^{--}, H^{+-}, H^{-+}, H^{++}\}$ (c) by splitting along each possible dimension. These sub-cubes are then re-combined to form two possible split candidates $\{H^{1-}, H^{1+}\}$ (d) and $\{H^{2-}, H^{2+}\}$ (e). Informed belief values are computed for the re-combined hypercubes, including a new estimate for $\hat{b}(H)$ (f), by summing the beliefs in the finer-scale partitioning. The new estimates are more accurate since the error introduced by the discretization decreases as the partitions become smaller.

these the *natural* weights for $p(x)$, given the $V_k$. There is a simple relationship between the quality of a *naturally-weighted* discretization and its entropy $\mathbb{H}(\cdot)$:

**Theorem 1.** *Among any collection of naturally-weighted discretizations of $p(x)$, the minimum KL divergence to $p(x)$ is achieved by a discretization of minimal entropy.*

*Proof.* For a naturally-weighted discretization $q$, $\mathrm{KL}(p||q) = -\sum_{k=1}^{K} w_k \log \frac{w_k}{|V_k|} + \int_U p \log p = \mathbb{H}(q) - \mathbb{H}(p)$. $\mathbb{H}(p)$ is constant, so $\mathrm{KL}(p||q)$ is minimized by minimizing $\mathbb{H}(q)$. □

Suppose we are given a discretization $\{H_\alpha^k\}$ and have computed messages and beliefs for every node using (2)–(4). The messages have not necessarily reached a fixed point, but we nevertheless have some current estimate for them. For any arbitrary hypercube $H$ at $x_\alpha$ (not necessarily in its current discretization) we can define the *informed belief*, denoted $\hat{b}(H)$, to be the belief $H$ would receive if all other nodes and their incoming messages were left unaltered. To compute the informed belief, one first computes new discrete factor function values involving $H$ using integrals like (1). These values are fed into (2), (3) to produce "informed" messages $m_{i,\alpha}(H)$ arriving at $x_\alpha$ from each neighbor $f_i$. Finally, the informed messages are fed into (4) to obtain the informed belief $\hat{b}(H)$.

## 3 Continuously-adaptive discretization

The core of the CAD-MP algorithm is the procedure for passing a message to a variable $x_\alpha$. Given fixed approximations at every other node, any discretization of $\alpha$ induces an approximate belief distribution $q_\alpha(x_\alpha)$. The task of the algorithm is to select the best discretization, and as Theorem 1 shows, a good strategy for this selection is to look for a naturally-weighted discretization that minimizes the entropy of $q_\alpha$. We achieve this using a new algorithm called "informed splitting" which is described next.

CAD-MP employs an axis-aligned binary-split kd-tree [15] to represent the discrete partitioning of a $D$-dimensional continuous state space at each variable (the same representation was used in [14] where it was called a Binary Split Partitioning). For our purposes, a kd-tree is a binary tree in which each vertex is assigned a subset—actually a hypercube—of the state space. The root is assigned the whole space, and any internal vertex splits its hypercube equally between its two children using an axis-aligned plane. The subsets assigned to all leaves partition the state space into hypercubes.

We build the kd-tree greedily by recursively splitting leaf vertices: at each step we must choose a hypercube $H_\alpha^k$ in the current partitioning to split, and a dimension $d$ to split it. According to Theorem 1, we should choose $k$ and $d$ to minimize the entropy of the resulting discretization—provided that this discretization has "natural" weights. In practice, the natural weights are estimated using informed beliefs; we nevertheless proceed as though they were exact and choose the $k$- and

$d$-values leading to lowest entropy. A subroutine of the algorithm involves "expanding" a hypercube into sub-cubes as illustrated in the two-dimensional case in Figure 1. The expansion procedure generalizes to $D$ dimensions by first expanding to $2^D$ subcubes and then re-combining these into $2D$ candidate splits. Note that for all $d \in \{1, \ldots, D\}$

$$\hat{b}(H) \equiv \hat{b}(H^{d-}) + \hat{b}(H^{d-}). \tag{5}$$

Once we have expanded each hypercube in the current partitioning and thereby computed values for $\hat{b}(H_\alpha^k)$, $\hat{b}(H_\alpha^{k,d-})$ and $\hat{b}(H_\alpha^{k,d+})$ for all $k$ and $d$, we choose $k$ and $d$ to minimize the "split entropy"

$$\gamma_\alpha(k, d) = -\sum_{i \neq k} \hat{b}(H_\alpha^i) \log \frac{\hat{b}(H_\alpha^i)}{|H_\alpha^i|} - \hat{b}(H_\alpha^{k,d-}) \log \frac{\hat{b}(H_\alpha^{k,d-})}{|H_\alpha^{k,d-}|} - \hat{b}(H_\alpha^{k,d+}) \log \frac{\hat{b}(H_\alpha^{k,d+})}{|H_\alpha^{k,d+}|}. \tag{6}$$

Note that from (5) we can perform this minimization without normalizing the $\hat{b}(\cdot)$.

We can now describe the CAD-MP algorithm using informed splitting, which re-partitions a variable of the factor graph by producing a new kd-tree whose leaves are the hypercubes in the new partitioning:

1. Initialize the root vertex of the kd-tree with its associated hypercube being the whole state space, with belief 1. Add this root to a leaf set $\mathcal{L}$ and "expand" it as shown in Figure 1.

2. While the number of leaves $|\mathcal{L}|$ is less than the desired number of partitions in the discretized model:
   (a) Pick the leaf $H$ and split dimension $d$ that minimize the split-entropy (6).
   (b) Create two new vertices $H^-$ and $H^+$ by splitting $H$ along dimension $d$, and "expand" these new vertices.
   (c) Remove $H$ from $\mathcal{L}$, and add $H^-$ and $H^+$ to $\mathcal{L}$.

All variables in the factor graph are initialized with the trivial discretization (a single partition). Variables can be visited according to any standard message-passing schedule, where a "visit" consists of repartitioning according to the above algorithm. A simple example showing the evolution of the belief at one variable is shown in Figure 2.

If the variable being repartitioned has $T$ neighbors and we require a partitioning of $K$ hypercubes, then a straightforward implementation of this algorithm requires the computation of $2K \times 2^D \times KT$ message components. Roughly speaking, then, informed splitting pays a factor of $2^{D+1}$ over BP which must compute $K^2T$ message components. But CAD-MP trades this for an exponential factor in $K$ since it can home in on interesting areas of the state space using binary search, so if BP requires $K$ partitions for a given level of accuracy, CAD-MP (empirically) achieves the same accuracy with only $O(\log K)$ partitions. Note that in special cases, including some low-level vision applications [16], classical BP can be performed in $O(KT)$ time and space; however this is still prohibitive for large $K$.

## 4    Experiments

We would like to compare our candidate algorithms against the marginal belief distributions that would be computed by exact inference, however no exact inference algorithm is known for our models. Instead, for each experiment we construct a fine-scale uniform discretization $\mathcal{D}_f$ of the model and input data, and compute the marginal belief distributions $p(x_\alpha; \mathcal{D}_f)$ at each variable $x_\alpha$ using the standard forward-backward BP algorithm. Given a candidate approximation $\mathcal{C}$ we can then compare the marginals $p(x_\alpha; \mathcal{C})$ under that approximation to the fine-scale discretization by computing the KL-divergence $KL(p(x_\alpha; \mathcal{D}_f)||p(x_\alpha; \mathcal{C}))$ at each variable. In results below, we report the mean of this divergence across all variables in the graph, and refer to it in the text as $\mu(\mathcal{C})$. While a "fine-enough" uniform discretization will tend to the true marginals, we do not *a priori* know how fine that is. We therefore construct a sequence of coarser uniform discretizations $\mathcal{D}_c^i$ of the same model and data, and compute $\mu(\mathcal{D}_c^i)$ for each of them. If $\mu(\mathcal{D}_c^i)$ is converging rapidly enough to zero, as is the case in the experiments below, we have confidence that the fine-scale discretization is a good approximation to the exact marginals.

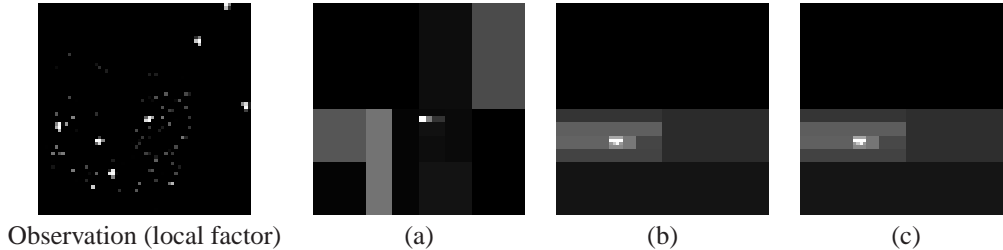

| Observation (local factor) | (a) | (b) | (c) |

Figure 2: **Evolution of discretization at a single variable.** The left image is the local (single-variable) factor at the first node in a simple chain MRF whose nodes have 2-D state spaces. The next three images, from left to right, show the evolution of the informed belief. Initially (a) the partitioning is informed simply by the local factor, but after messages have been passed once along the chain and back (b), the posterior marginal estimate has shifted and the discretization has adapted accordingly. Subsequent iterations over the chain (c) do not substantially alter the estimated marginal belief. For this toy example only 16 partitions are used, and the normalized log of the belief is displayed to make the structure of the distribution more apparent.

We compare our adaptive discretization algorithm against non-parametric belief propagation (NBP) [9, 10] which represents the marginal distribution at a variable by a particle set. We generate some importance samples directly from the observation distribution, both to initialize the algorithm and to "re-seed" the particle set when it gets lost. Particle sets typically do not approximate the tails of a distribution well, leading to zeros in the approximate marginals and divergences that tend to infinity. We therefore regularize all divergence computations as follows:

$$KL^*(p||q) = \sum_k p_k^* \log(\frac{p_k^*}{q_k^*}), \quad p_k^* = \frac{\epsilon + \int_{H^k} p(x)}{\sum_n (\epsilon + \int_{H^n} p(x))}, \quad q_k^* = \frac{\epsilon + \int_{x_k} q(x)}{\sum_n (\epsilon + \int_{H^n} q(x))} \quad (7)$$

where $\{H^k\}$ are the partitions in the fine-scale discretization $\mathcal{D}_f$. All experiments use $\epsilon = 10^{-4}$ which was found empirically to show good results for NBP.

We begin with a set of experiments over ten randomly generated input sequences of a one-dimensional target moving through structured clutter of similar-looking distractors. One of the sequences is shown in Figure 3a, where time goes from bottom to top. The measurement at a time-step consists in 240 "pixels" (piecewise-constant regions of uniform width) generated by simulating a small one-dimensional target in clutter, with additive Gaussian shot-noise. There are stationary clutter distractors, and also periodic "forkings" where a moving clutter distractor emerges from the target and proceeds for a few time-steps before disappearing. Each sequence contains 256 time-steps, and the "exact" marginals (Figure 3b) are computed using standard discrete BP with 15360 states per time-step. The modes of the marginals generated by all the experiments are similar to those in Figure 3b, except for one run of NBP shown in Figure 3c that failed entirely to find the mode (red line) due to an unlucky random seed. However, the distributions differ in fine structure, where CAD-MP approximates the tails of the distribution much better than NBP.

Figure 4a shows the divergences $\mu(\cdot)$ for the various discrete algorithms: both uniform discretization at various degrees of coarseness, and adaptive discretization using CAD-MP with varying numbers of partitions. Each data point shows the mean divergence $\mu(\cdot)$ for one of the ten simulated one-dimensional datasets. As the number of adaptive partitions increases, the variance of $\mu(\cdot)$ across trials increases, but the divergence stays small. Higher divergences in CAD-MP trials correspond to a mis-estimation of the tails of the marginal belief at a few time-steps. The straight line on the log/log plot for the uniform discretizations gives us confidence that the fine-scale discretization is a close approximation to the exact beliefs. The adaptive discretization provides a very faithful approximation to this "exact" distribution with vastly fewer partitions.

Figure 4b shows the divergences for the same ten one-dimensional trial sequences when the marginals are computed using NBP with varying numbers of particles. The NBP algorithm was run five times on each of the ten simulated one-dimensional datasets with different random seeds each time, and the particle-set sizes were chosen to approximately match the computation time of the CAD-MP algorithm. The NBP algorithm does worse absolutely (the divergences are much larger even after regularization, indicating that areas of high belief are sometimes mis-estimated), and also

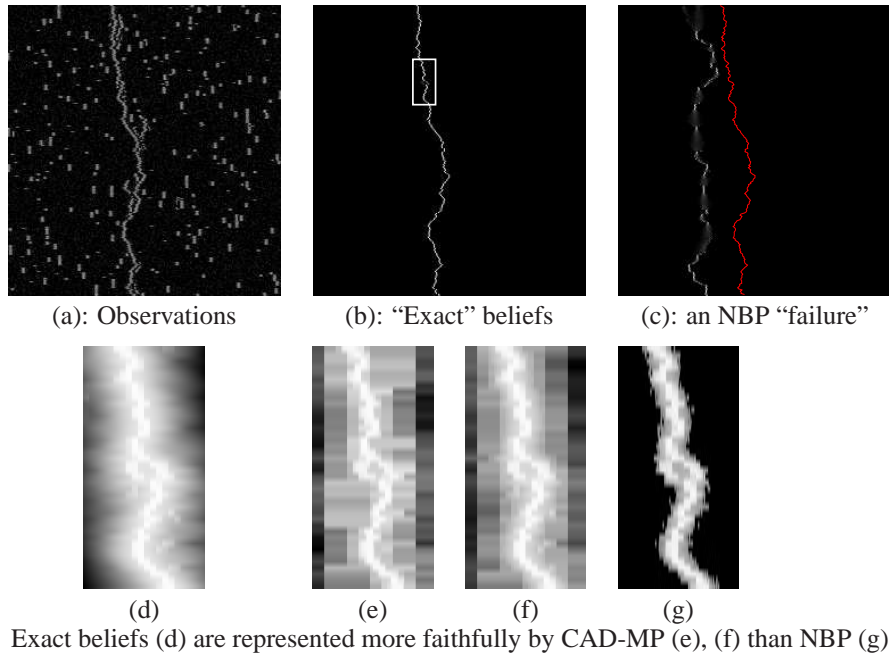

(a): Observations      (b): "Exact" beliefs      (c): an NBP "failure"

(d)      (e)      (f)      (g)

Exact beliefs (d) are represented more faithfully by CAD-MP (e), (f) than NBP (g)

Figure 3: **One of the one-dimensional test sequences.** The region of the white rectangle in (b) is expanded in (d)–(g), with beliefs now plotted on log intensity scale to expand their dynamic range. CAD-MP using only 16 partitions per time-step (e) already produces a faithful approximation to the exact belief (d), and increasing to 128 partitions (f) fills in more details. The NBP algorithm using 800 particles (g) does not approximate the tails of the distribution well.

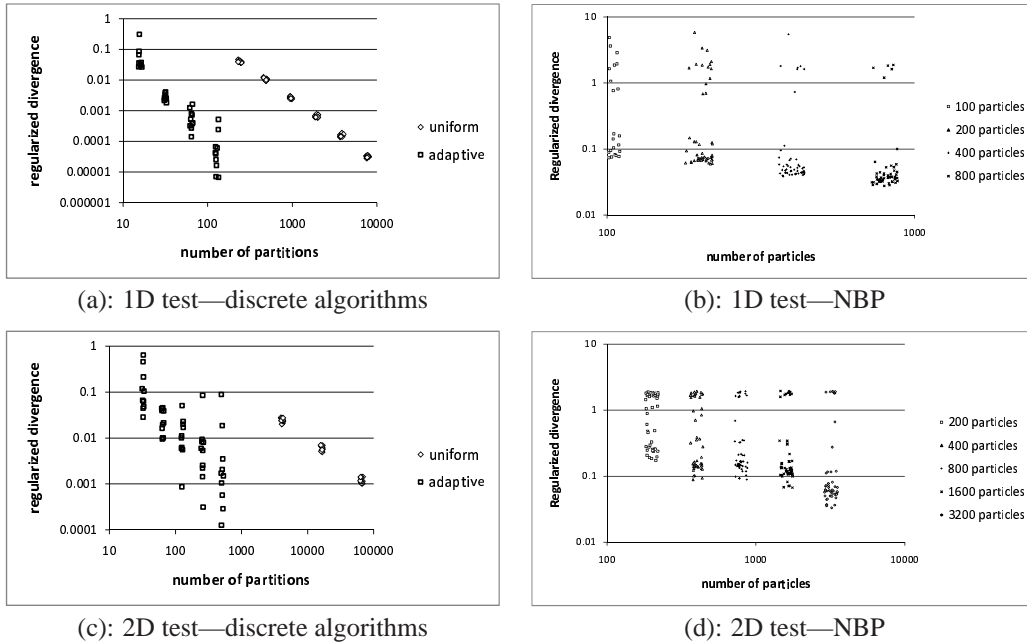

(a): 1D test—discrete algorithms      (b): 1D test—NBP

(c): 2D test—discrete algorithms      (d): 2D test—NBP

Figure 4: **Adaptive discretization achieves the same accuracy as uniform discretization using many fewer partitions, but non-parametric belief propagation is less effective.** See Section 4 for details.

varies greatly across different trial sequences, and when re-run with different random seeds on the same trial sequence. Note also that the $\mu(\cdot)$ are bi-modal—values of $\mu(\cdot)$ above around 0.5 signify runs on which NBP incorrectly located the mode of the marginal belief distribution at some or all time-steps, as in Figure 3c.

We performed a similar set of experiments using a simulated two-dimensional data-set. This time the input data is a $64 \times 64$ image grid, and the "exact" fine-scale discretization is at a resolution of $512 \times 512$ giving $262144$ discrete states in total. Figures 4c and 4d show that adaptive discretization still greatly outperforms NBP for an equivalent computational cost. Again there is a straight-line trend in the log/log plots for both CAD-MP and uniform discretization, though as in the one-dimensional case the variance of the divergences increases with more partitions. NBP again performs less accurately, and frequently fails to find the high-weight regions of the belief at all at some time-steps, even with 3200 particles.

Adaptive discretization seems to correct some of the well-known limitations of particle-based methods. The discrete distribution is able to represent probability mass well into the tails of the distribution, which leads to a more faithful approximation to the exact beliefs. This also prevents the catastrophic failure case for NBP shown in Figure 3c, where the mode of the distribution is lost entirely because no particles were placed nearby. Moreover, CAD-MP's computational complexity scales linearly with the number of incoming messages at a factor. NBP has to resort to heuristics to sample from the product of incoming messages once the number of messages is greater than two.

## 5   Related work

The work most closely related to CAD-MP is the 1997 algorithm of Kozlov and Koller [14]. We refer to this algorithm as "KK97"; its main differences to CAD-MP are: (i) KK97 is described in a junction tree setting and computes the marginal posterior of just the root node, whereas CAD-MP computes beliefs everywhere in the graph; (ii) KK97 discretizes *messages* (on junction tree edges) rather than *variables* (in a factor graph), so multiplying incoming messages together requires the substantial additional complexity of merging disparate discretizations, compared to CAD-MP in which the incoming messages share the same discretization. Difference (i) is the more serious, since it renders KK97 inapplicable to the type of early-vision problem we are motivated by, where the marginal at every variable must be estimated.

Coarse-to-fine techniques can speed up the convergence of loopy BP [16] but this does not address the discrete state-space explosion. One can also prune the state space based on local evidence [17, 18]. However, this approach is unsuitable when the data function has high entropy; moreover, it is very difficult to bring a state back into the model once it has been pruned.

Another interesting approach is to retain the uniform discretization, but enforce sparsity on messages to reduce computational cost. This was done in both [19] (in which messages are approximated using a using a mixture of delta functions, which in practice results in retaining the $K$ largest message components) and [20] (which uses an additional uniform distribution in the approximating distribution to ensure non-zero weights for all states in the discretization). However, these approaches appear to suffer when multiplying messages with disjoint peaks whose tails have been truncated to enforce sparsity: such peaks are unable to fuse their evidence correctly. Also, [20] is not directly applicable when the state-space is multi-dimensional.

Expectation Propagation [5] is a highly effective algorithm for inference in continuous-valued networks, but is not valid for densities that are multimodal mixtures.

## 6   Discussion

We have demonstrated that our new algorithm, CAD-MP, performs accurate approximate inference with complex, multi-modal observation distributions and corresponding multi-modal posterior distributions. It substantially outperforms the two standard methods for inference in this setting: uniform-discretization and non-parametric belief propagation. While we only report results here on simulated data, we have successfully used the method on low-level vision problems and are preparing a companion publication to describe these results. We believe CAD-MP and variants on it may be applicable to other domains where complex distributions must be estimated in spaces of low to

moderate dimension. The main challenge in applying the technique to an arbitrary factor graph is the tractability of the definite integrals (1).

This paper describes a particular set of engineering choices motivated by our problem domain. We use kd-trees to describe partitionings: other data structures could certainly be used. Also, we employ a greedy heuristic to select a partitioning with low entropy rather than exhaustively computing a minimimum entropy over some family of discretizations. We have experimented with a Metropolis algorithm to augment this greedy search: a Metropolis move consists in "collapsing" some sub-tree of the current partitioning and then re-expanding using a randomized form of the minimum-entropy criterion. We have also tried tree-search heuristics that do not need the $O(2^D)$ "expansion" step, and thus may be more effective when $D$ is large. The choices reported here seem to give the best accuracy on our problems for a given computational budget, however many others are possible and we hope this work will serve as a starting point for a renewed interest in adaptive discretization in a variety of inference settings.

# References

[1] J. Pearl. *Probabilistic Reasoning in Intelligent Systems: Networks of Plausible Inference*. Morgan Kaufmann, 1988.

[2] P. Dagum and M. Luby. Approximating probabilistic inference in bayesian belief networks is NP-hard. *Artificial Intelligence*, 60(1):141–153, 1993.

[3] Robert G. Cowell, A. Philip Dawid, Steffen L. Lauritzen, and David J. Spiegelhalter. *Probabilistic Networks and Expert Systems*. Springer, 1999.

[4] Jonathan S. Yedidia, William T. Freeman, and Yair Weiss. Generalized belief propagation. In *NIPS*, pages 689–695, 2000.

[5] T. Minka. Expectation propagation for approximate bayesian inference. In *Proc. UAI*, pages 362–369, 2001.

[6] G. Kitagawa. The two-filter formula for smoothing and an implementation of the gaussian-sum smoother. *Ann. Inst. Statist. Math.*, 46(4):605–623, 1994.

[7] P.F. Felzenszwalb and D.P. Huttenlocher. Efficient belief propagation for early vision. In *Proc. CVPR*, 2004.

[8] M. Isard and J. MacCormick. Dense motion and disparity estimation via loop belief propagation. In *ACCV*, pages 32–41, 2006.

[9] E. Sudderth, A. Ihler, W. Freeman, and A. Willsky. Nonparametric belief propagation. In *Proc. CVPR*, volume 1, pages 605–612, 2003.

[10] M. Isard. Pampas: Real-valued graphical models for computer vision. In *Proc. CVPR*, volume 1, pages 613–620, 2003.

[11] F.R. Kschischang, B.J. Frey, and H.A. Loeliger. Factor graphs and the sum-product algorithm. *IEEE Transactions on Information Theory*, 47(2):498–519, 2001.

[12] O. Zoeter and H. Heskes. Deterministic approximate inference techniques for conditionally gaussian state space models. *Statistics and Computing*, 16(3):279–292, 2006.

[13] T. Minka. Divergence measures and message passing. Technical Report MSR-TR-2005-173, Microsoft Research, 2005.

[14] Alexander V. Kozlov and Daphne Koller. Nonuniform dynamic discretization in hybrid networks. In *Proc. UAI*, pages 314–325, 1997.

[15] Jon Louis Bentley. Multidimensional binary search trees used for associative searching. *Commun. ACM*, 18(9):509–517, 1975.

[16] P.F. Felzenszwalb and D.P. Huttenlocher. Pictorial structures for object recognition. *Int. J. Computer Vision*, 61(1):55–79, 2005.

[17] J. Coughlan and S. Ferreira. Finding deformable shapes using loopy belief propagation. In *Proc. ECCV*, pages 453–468, 2002.

[18] J. Coughlan and H. Shen. Shape matching with belief propagation: Using dynamic quantization to accommodate occlusion and clutter. In *Proc. Workshop on Generative-Model Based Vision*, 2004.

[19] C. Pal, C. Sutton, and A. McCallum. Sparse forward-backward using minimum divergence beams for fast training of conditional random fields. In *International Conference on Acoustics, Speech, and Signal Processing*, 2006.

[20] J. Lasserre, A. Kannan, and J. Winn. Hybrid learning of large jigsaws. In *Proc. CVPR*, 2007.

